# An Analog VLSI Splining Network

**Daniel B. Schwartz and Vijay K. Samalam**
GTE Laboratories, Inc.
40 Sylvan Rd.
Waltham, MA 02254

## Abstract

We have produced a VLSI circuit capable of learning to approximate arbitrary smooth of a single variable using a technique closely related to splines. The circuit effectively has 512 knots space on a uniform grid and has full support for learning. The circuit also can be used to approximate multi-variable functions as sum of splines.

An interesting, and as of yet, nearly untapped set of applications for VLSI implementation of neural network learning systems can be found in adaptive control and non-linear signal processing. In most such applications, the learning task consists of approximating a real function of a small number of continuous variables from discrete data points. Special purpose hardware is especially interesting for applications of this type since they generally require real time on-line learning and there can be stiff constraints on the power budget and size of the hardware. Frequently, the already difficult learning problem is made more complex by the non-stationary nature of the underlying process.

Conventional feed-forward networks with sigmoidal units are clearly inappropriate for applications of this type. Although they have exhibited remarkable performance in some types of time series prediction problems (for example, Wiegend, 1990 and Atlas, 1990), their learning rates in general are too slow for on-line learning. On-line performance can be improved most easily by using networks with more constrained architecture, effectively making the learning problem easier by giving the network a hint about the learning task. Networks that build local representations of the data, such as radial basis functions, are excellent candidates for these type of problems. One great advantage of such networks is that they require only a single layer of units. If the position and width of the units are fixed, the learning problem is linear

in the coefficients and local. By local we mean the computation of a weight change requires only information that is locally available to each weight, a highly desirable property for VLSI implementation. If the learning algorithm is allowed to adjust both the position and width of the units then many of the advantages of locally tuned units are lost.

A number of techniques have been proposed for the determination of the width and placement of the units. One of the most direct is to center a unit at every data point and to adjust the widths of the units so the receptive fields overlap with those of neighboring data points ( Broomhead, 1989 ). The proliferation of units can be limited by using unsupervised clustering techniques to clump the data followed by the allocation of units to fit the clumps (Moody, 1989). Others have advocated assigning new units only when the error on a new data point is larger than a threshold and otherwise making small adjustments in the weights and parameters of the existing units (Platt, 1990). All of these methods suffer from the common problem of requiring an indeterminate quantity of resources in contrast with the fixed resources available from most VLSI circuits. Even worse, when used with non-stationary processes a mechanism is needed to deallocate units as well as to allocate them. The resource allocation/deallocation problem is a serious barrier to implementing these algorithms as autonomous VLSI microsystems.

## A Splining Network

To avoid the resource allocation problem we propose a network that uses all of its weights and units regardless of the problem. We avoid over parameterization of the training data by building constraints on smoothness into the network, thus reducing the number of degrees of freedom available to the training process. In its simplest guise, the network approximates arbitrary 1-d smooth functions with a linear superposition of locally tuned units spaced on a uniform grid,

$$g(x) = \sum_i \omega_i f_\sigma(x - i\Delta x) \tag{1}$$

where $\sigma$ is the radius of the unit's receptive field and the $\omega_i$ are the weights. $f_\sigma$ is a bump of width $\sigma$ such as a gaussian or a cubic spline basis function. Mathematically the network is closely related to function approximation using B-splines (Lancaster, 1986) with uniformly spaced knots. However, in B-spline interpolation the overlap of the basis functions is normally determined by the degree of the spline whereas we use the degree of overlap as a free parameter to constrain the smoothness of the network's output. As mentioned earlier, the network is linear in its weights so gradient descent with a quadratic cost function (LMS) is an effective training procedure.

The weights needed for this network can easily be implemented in CMOS with an array of transconductance amplifiers. The amplifiers are wired as voltage followers with their outputs tied together and the weights are represented by voltages $V_i$ at the non-inverting inputs of the amplifiers. If the outputs of the locally tuned units are represented by unipolar currents $I_i$ these currents can be used to bias the

transconductance amplifiers and the result is (Mead,1989)

$$V_{out} = \frac{\sum_i I_i V_i}{\sum_i I_i}$$

provided that care is taken to control the non-linearities of the amplifiers. However, while the weights have a simple implementation in analog VLSI circuitry, the input units do not. A number of circuits exist whose transfer characteristics can be shaped to be a suitable bump but none of those known to the authors allow the width of the bump to be adjusted over a wide range without the use of resistors.

## Generating the Receptive Fields

Input units with tunable receptive fields can be generated quite efficiently by breaking them up into two layers of circuitry as shown in figure 1. The input layer place encodes the input signal – i.e. only one or perhaps a small cluster of units is active at a time. The output of the place encoding units either injects or controls the

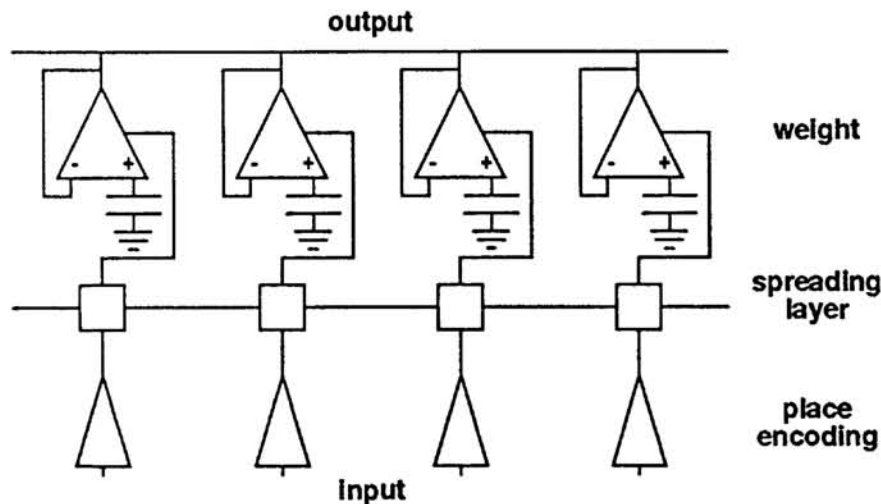

Figure 1: An architecture that allows the width and shape of the receptive fields to be varied over a wide range. The elements of the 'spreading layer' are passive and can sink current to ground.

injection of current into the laterally connected spreading layer. The elements in the spreading layer all contain ground terminals and the current sunk by each one determines the bias current applied to the associated weight. Clearly, the distribution of currents flowing to ground through the spreading layer form a smooth bump such that when excitation is applied to tap j of the spreading layer,

$$I_i = I_o f_\sigma(i - j)$$

where $f_\sigma(i)$ is the bump called for by equation 1. In our earliest realizations of this network the input layer was a crude flash A-to-D converter and the input to the circuit was analog. In the current generation the input is digital with the place encoding performed by a conventional address decoder. If desired, input quantization can be avoided by using a layer of amplifiers that generate smooth bumps of fixed width to generate the input place encoding.

The simplest candidate to implement the spreading layer in conventional CMOS is a set of diode connected n-channel transistors laterally connected by n-channel pass transistors. The gate voltages of the diode connected transistors determine the bias currents $I_i$ of the weights. Ignoring the body effect and assuming weak inversion in the current sink, this type of networks tends to gives bumps with rather sharp peaks, $I_i \approx \sum_j I_o e^{-\alpha|j|}$, where $|j|$ is the distance from the point where the excitation is applied. Figure 2 shows a more sophisticated version of this circuit in which the output of the place encoding units applies excitation to the spreading network through a p-channel transistor. The shape of the bumps can be softened by

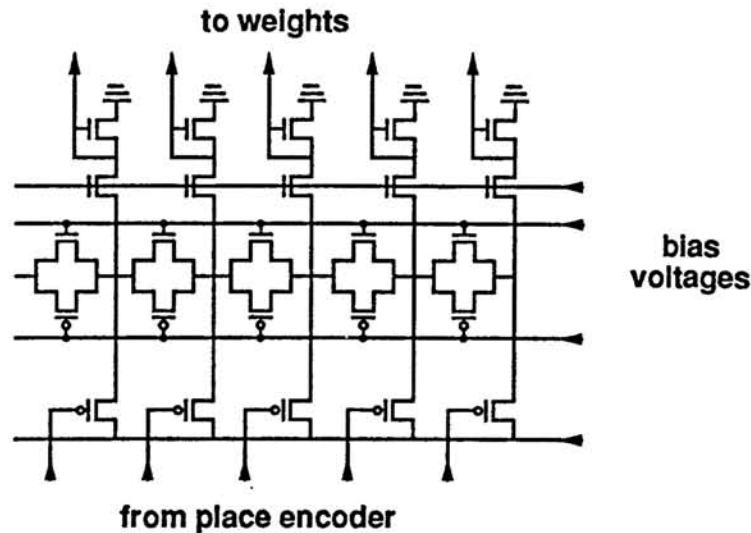

Figure 2: A schematic of a section of the spreading layer. Roughly speaking, the n-channel pass transistor controls the extent of the tails of the bumps and the p-channel pass transistor and the cascode transistor control its width.

limiting the amount of current drawn by the current sinks with an n-channel cascode transistor in series with the current sink. Some experimental results for this type of circuit are shown in figure 3a. More control can be obtained by using complementary pass transistors. The use of p-channel pass transistors alone unexpectedly results in bumps that are nearly square (figure 3b). These can be smoothed by using a using both flavors of pass transistor simultaneously (figure 3c).

## The Weights

As described earlier, the implementation of the output weights is based on the computation of means by the well known follower-aggregation circuit. With typical transconductance amplifiers, this averaging is linear only when the voltages being averaged are distributed over a voltage range of no more than a few time $U_Q = kT/e$ in weak inversion. In the circuits described here the linear range has been widened to nearly a volt by reducing the transconductance of the readout amplifiers through the combination of low width to length ratio input transistors and relatively large tail currents.

The weights $V_i$ are stored on MOS capacitors and are programmed by the gated transconductance amplifier shown in figure 4. Since this amplifier computes the

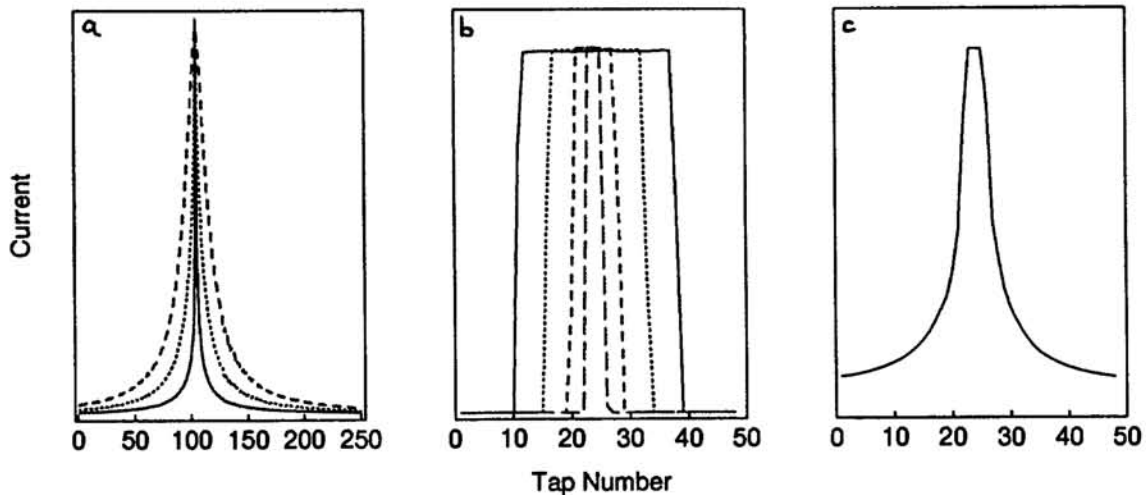

Figure 3: Experimental measurements of the receptive field shapes obtained from different types of networks. (a) n-channel transistors for several gate voltages. (b) p-channel transistors for several gate voltages. (c) Both n-channel and p-channel pass transistors.

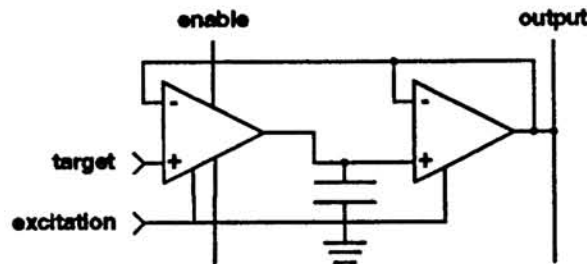

Figure 4: Schematic of an output weight including the circuitry to generate weight updates. To minimize leakage and charge injection simultaneously, the pass transistors used to gate the weight change amplifier are of minimum size and a separate transistor turns off the output transistors of the amplifier.

difference between the target voltage and the actual output of the network, the learning rule is just LMS,

$$\Delta V_i = \frac{g_i \tau}{C}(V_{target} - V_{out}), \ \Delta V_i << V_{target} - V_{out}$$

where C is the capacitance of the storage capacitor and $\tau$ is the duration of weight changes. The transconductance $g_i$ of the weight change amplifier is determined by the strength of excitation current from the spreading layer, $g_i \propto I_i$ in weak inversion. Since the weight changes are governed by strengths of the excitation currents from the spreading layer, clusters of weights are changed at a time. This enhances the fault tolerance of the circuit since the group of weights surrounding a bad one can compensate for it.

## Experimental Evaluation

Several different chips have been fabricated in $2\mu$ p-well CMOS and tested to evaluate the principles described here. The most recent of these has 512 weights arranged in a $64 \times 8$ matrix connected to form a one dimensional array. The active area of this chip is $4.1mm \times 3.7mm$. The input signal is digital with the place encoding performed by a conventional address decoder. To maximize the flexibility of the chip, the excitation is applied to the spreading layer by a register located in each cell. By writing to multiple registers between resets, the spreading layer can be excited at multiple points simultaneously. This feature allows the chip to be treated as a single 1-dimensional spline with 512 weights or, for example, as the sum of four distinct 1-dimensional splines each made up of 128 weights. One of the most noticeable virtues of this design is the simplicity of the layout due to the absence of any clear distinction between 'weights' and 'units'. The primitive cell consists of a register, a piece of the spreading network, a weight change amplifier, a storage capacitor and output amplifier. All but a tiny fraction of the chip is a tiling of this primitive cell. The excess circuitry consists of the address decoders, a timing circuit to control the duration of weight changes and some biasing circuitry for the spreading layer.

To execute LMS learning, the user need only provide a sequence of target voltages and a current proportional to the duration of weight changes. Under reasonable operating conditions a weight updates cycle takes less than $1\mu s$ implying a weight change rate of $5 \times 10^8$ connections/second. The response of the chip to a single weight change after initialization is shown in in figure 5a. One feature of this plot is striking – even though the distribution of offsets in the individual amplifiers has a variance of $13mV$, the ripple in the output of the chip is about a $1mV$. For some computations, it appears the limiting factor on the accuracy of the chip is the rate of weight decay, about $10mV/s$.

As a more strenuous test of the functionality of the chip we trained it to predict chaotic time series generated by the well know logistic equation,

$$x_{t+1} = 4\alpha x_t(1 - x_t), \ \alpha < 1.$$

Some experimental results for the mean prediction error are shown in figure 5b. In these experiments, a mean prediction error of 3% is achieved, which is well above the intrinsic accuracy of the circuit. A detailed examination of the error rate as a function of the size and shape of the bumps indicates that the problem lies in the long tails exhibited by the spreading layer when the n-channel pass transistors are turned on. This tail falls off very slowly due to the body effect. One remedy to this problem is to actively bias the gates of the n-channel pass transistors to be a programmed offset above their source voltages (Mead, 1989). A simpler solution is to subtract a fixed current from each of the bias current defined by the spreading layer. This solution costs a mere 4 transistors and has the added benefit of guaranteeing that the bumps will always have a finite support.

## Conclusion

We have demonstrated that neural network learning can be efficiently mapped onto analog VLSI provided that the network architecture and training procedure are

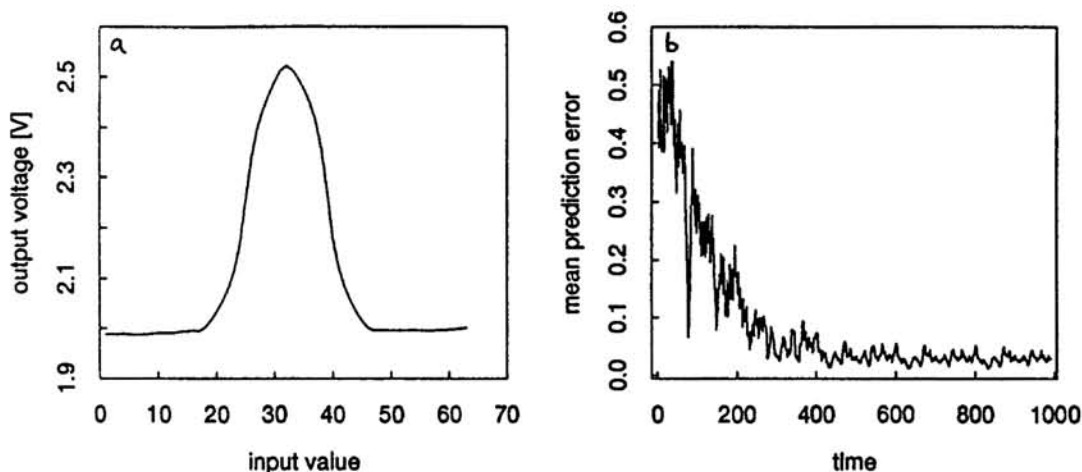

Figure 5: Some experimental results from a splining circuit. (a) The response of the circuit to learning one data point after initialization of the weights to a constant value. (b) Experimental mean prediction while learning a chaotic time series.

tailored to match the constraints imposed by VLSI. Besides the computational speed and low power consumption ( $300\mu A$ ) that follow directly from this mapping onto VLSI, the circuit also demonstrates intrinsic fault tolerance to defects in the weights.

## Acknowledgements

This work was initially inspired by a discussion with A. G. Barto and R. S. Sutton. A discussion with J. Moody was also helpful.

## References

[1] L. Atlas, R. Cole, Y. Muthusamy, A. Lippman, J. Connor, D. Park, M. El-Sharkawi, and R. J. Marks II. A performance comparison of trained multi-layer perceptrons and trained classification trees. *IEEE Proceedings*, 1990.

[2] D. S. Broomhead and D. Lowe. Multivariable function interpolation and adaptive networks. *Complex Systems*, 2:321-355, 1988.

[3] P. Lancaster and K. Šalkauskas. *Curve and Surface Fitting*. Academic Press, 1986.

[4] C. Mead. *Analog VLSI and Neural Systems*. Addison-Wesley, 1989.

[5] J. Moody and C.J. Darken. Fast learning in networks of locally-tuned processing units. *Neural Computation*, 1(2), 1989.

[6] J. Platt. A resource-allocating neural network for function interpolation. In Richard P. Lippman, John Moody, and David S. Touretzky, editors, *Advances in Neural Information Processing Systems 3*, 1991.

[7] A. S. Weigend, , B. A. Huberman, and D. E. Rummlehart. Predicting the future : A connectionist approach. *International Journal of Neural Systems*, 3, 1990.